# Large Margin Hidden Markov Models for Automatic Speech Recognition

**Fei Sha**
Computer Science Division
University of California
Berkeley, CA 94720-1776
feisha@cs.berkeley.edu

**Lawrence K. Saul**
Department of Computer Science and Engineering
University of California (San Diego)
La Jolla, CA 92093-0404
saul@cs.ucsd.edu

## Abstract

We study the problem of parameter estimation in continuous density hidden Markov models (CD-HMMs) for automatic speech recognition (ASR). As in support vector machines, we propose a learning algorithm based on the goal of margin maximization. Unlike earlier work on max-margin Markov networks, our approach is specifically geared to the modeling of real-valued observations (such as acoustic feature vectors) using Gaussian mixture models. Unlike previous discriminative frameworks for ASR, such as maximum mutual information and minimum classification error, our framework leads to a convex optimization, without any spurious local minima. The objective function for large margin training of CD-HMMs is defined over a parameter space of positive semidefinite matrices. Its optimization can be performed efficiently with simple gradient-based methods that scale well to large problems. We obtain competitive results for phonetic recognition on the TIMIT speech corpus.

## 1 Introduction

As a result of many years of widespread use, continuous density hidden Markov models (CD-HMMs) are very well matched to current front and back ends for automatic speech recognition (ASR) [21]. Typical front ends compute real-valued feature vectors from the short-time power spectra of speech signals. The distributions of these acoustic feature vectors are modeled by Gaussian mixture models (GMMs), which in turn appear as observation models in CD-HMMs. Viterbi decoding is used to solve the problem of *sequential classification* in ASR—namely, the mapping of sequences of acoustic feature vectors to sequences of phonemes and/or words, which are modeled by state transitions in CD-HMMs.

The simplest method for parameter estimation in CD-HMMs is the Expectation-Maximization (EM) algorithm. The EM algorithm is based on maximizing the *joint likelihood* of observed feature vectors and label sequences. It is widely used due to its simplicity and scalability to large data sets, which are common in ASR. A weakness of this approach, however, is that the model parameters of CD-HMMs are not optimized for sequential classification: in general, maximizing the joint likelihood does not minimize the phoneme or word error rates, which are more relevant metrics for ASR.

Noting this weakness, many researchers in ASR have studied alternative frameworks for parameter estimation based on conditional maximum likelihood [11], minimum classification error [4] and maximum mutual information [20]. The learning algorithms in these frameworks optimize *discriminative* criteria that more closely track actual error rates, as opposed to the EM algorithm for maximum likelihood estimation. These algorithms do not enjoy the simple update rules and relatively fast convergence of EM, but carefully and skillfully implemented, they lead to lower error rates [13, 20].

Recently, in a new approach to discriminative acoustic modeling, we proposed the use of "large margin GMMs" for multiway classification [15]. Inspired by support vector machines (SVMs), the learning algorithm in large margin GMMs is designed to maximize the distance between labeled examples and the decision boundaries that separate different classes [19]. Under mild assumptions, the required optimization is convex, without any spurious local minima. In contrast to SVMs, however, large margin GMMs are very naturally suited to problems in multiway (as opposed to binary) classification; also, they do not require the kernel trick for nonlinear decision boundaries. We showed how to train large margin GMMs as segment-based phonetic classifiers, yielding significantly lower error rates than maximum likelihood GMMs [15]. The integrated large margin training of GMMs and transition probabilities in CD-HMMs, however, was left as an open problem.

We address that problem in this paper, showing how to train large margin CD-HMMs in the more general setting of sequential (as opposed to multiway) classification. In this setting, the GMMs appear as acoustic models whose likelihoods are integrated over time by Viterbi decoding. Experimentally, we find that large margin training of HMMs for sequential classification leads to significant improvement *beyond* the frame-based and segment-based discriminative training in [15].

Our framework for large margin training of CD-HMMs builds on ideas from many previous studies in machine learning and ASR. It has similar motivation as recent frameworks for sequential classification in the machine learning community [1, 6, 17], but differs in its focus on the real-valued acoustic feature representations used in ASR. It has similar motivation as other discriminative paradigms in ASR [3, 4, 5, 11, 13, 20], but differs in its goal of margin maximization and its formulation of the learning problem as a convex optimization over positive semidefinite matrices. The recent margin-based approach of [10] is closest in terms of its goals, but entirely different in its mechanics; moreover, its learning is limited to the mean parameters in GMMs.

## 2   Large margin GMMs for multiway classification

Before developing large margin HMMs for ASR, we briefly review large margin GMMs for multiway classification [15]. The problem of multiway classification is to map inputs $\boldsymbol{x} \in \Re^d$ to labels $y \in \{1, 2, \ldots, C\}$, where $C$ is the number of classes. Large margin GMMs are trained from a set of labeled examples $\{(\boldsymbol{x}_n, y_n)\}_{n=1}^N$. They have many parallels to SVMs, including the goal of margin maximization and the use of a convex surrogate to the zero-one loss [19]. Unlike SVMs, where classes are modeled by half-spaces, in large margin GMMs the classes are modeled by collections of ellipsoids. For this reason, they are more naturally suited to problems in multiway as opposed to binary classification. Sections 2.1–2.3 review the basic framework for large margin GMMs: first, the simplest setting in which each class is modeled by a single ellipsoid; second, the formulation of the learning problem as a convex optimization; third, the general setting in which each class is modeled by two or more ellipsoids. Section 2.4 presents results on handwritten digit recognition.

### 2.1   Parameterization of the decision rule

The simplest large margin GMMs model each class by a single ellipsoid in the input space. The ellipsoid for class $c$ is parameterized by a centroid vector $\boldsymbol{\mu}_c \in \Re^d$ and a positive semidefinite matrix $\boldsymbol{\Psi}_c \in \Re^{d \times d}$ that determines its orientation. Also associated with each class is a nonnegative scalar offset $\theta_c \geq 0$. The decision rule labels an example $\boldsymbol{x} \in \Re^d$ by the class whose centroid yields the smallest Mahalanobis distance:

$$y = \operatorname*{argmin}_c \left\{ (\boldsymbol{x} - \boldsymbol{\mu}_c)^{\mathrm{T}} \boldsymbol{\Psi}_c (\boldsymbol{x} - \boldsymbol{\mu}_c) + \theta_c \right\}. \qquad (1)$$

The decision rule in eq. (1) is merely an alternative way of parameterizing the maximum a posterior (MAP) label in traditional GMMs with mean vectors $\boldsymbol{\mu}_c$, covariance matrices $\boldsymbol{\Psi}_c^{-1}$, and prior class probabilities $p_c$, given by $y = \operatorname{argmin}_c \{ p_c \mathcal{N}(\boldsymbol{\mu}_c, \boldsymbol{\Psi}_c^{-1}) \}$.

The argument on the right hand side of the decision rule in eq. (1) is nonlinear in the ellipsoid parameters $\boldsymbol{\mu}_c$ and $\boldsymbol{\Psi}_c$. As shown in [15], however, a useful reparameterization yields a simpler expression. For each class $c$, the reparameterization collects the parameters $\{\boldsymbol{\mu}_c, \boldsymbol{\Phi}_c, \theta_c\}$ in a single enlarged matrix $\boldsymbol{\Phi}_c \in \Re^{(d+1) \times (d+1)}$:

$$\boldsymbol{\Phi}_c = \left[ \begin{array}{cc} \boldsymbol{\Psi}_c & -\boldsymbol{\Psi}_c \boldsymbol{\mu}_c \\ -\boldsymbol{\mu}_c^{\mathrm{T}} \boldsymbol{\Psi}_c & \boldsymbol{\mu}_c^{\mathrm{T}} \boldsymbol{\Psi}_c \boldsymbol{\mu}_c + \theta_c \end{array} \right]. \qquad (2)$$

Note that $\mathbf{\Phi}_c$ is positive semidefinite. Furthermore, if $\mathbf{\Phi}_c$ is strictly positive definite, the parameters $\{\boldsymbol{\mu}_c, \mathbf{\Psi}_c, \theta_c\}$ can be uniquely recovered from $\mathbf{\Phi}_c$. With this reparameterization, the decision rule in eq. (1) simplifies to:

$$y = \underset{c}{\operatorname{argmin}} \left\{ \boldsymbol{z}^{\mathrm{T}} \mathbf{\Phi}_c \, \boldsymbol{z} \right\} \quad \text{where} \quad \boldsymbol{z} = \left[ \begin{array}{c} \boldsymbol{x} \\ 1 \end{array} \right]. \tag{3}$$

The argument on the right hand side of the decision rule in eq. (3) is linear in the parameters $\mathbf{\Phi}_c$. In what follows, we will adopt the representation in eq. (3), implicitly constructing the "augmented" vector $\boldsymbol{z}$ for each input vector $\boldsymbol{x}$. Note that eq. (3) still yields nonlinear (piecewise quadratic) decision boundaries in the vector $\boldsymbol{z}$.

## 2.2 Margin maximization

Analogous to learning in SVMs, we find the parameters $\{\mathbf{\Phi}_c\}$ that minimize the *empirical risk* on the training data—i.e., parameters that not only classify the training data correctly, but also place the decision boundaries as far away as possible. The margin of a labeled example is defined as its distance to the nearest decision boundary. If possible, each labeled example is constrained to lie at least one unit distance away from the decision boundary to each competing class:

$$\forall c \neq y_n, \quad \boldsymbol{z}_n^{\mathrm{T}}(\mathbf{\Phi}_c - \mathbf{\Phi}_{y_n}) \, \boldsymbol{z}_n \geq 1. \tag{4}$$

Fig. 1 illustrates this idea. Note that in the "realizable" setting where these constraints can be simultaneously satisfied, they do not uniquely determine the parameters $\{\mathbf{\Phi}_c\}$, which can be scaled to yield arbitrarily large margins. Therefore, as in SVMs, we propose a convex optimization that selects the "smallest" parameters that satisfy the large margin constraints in eq. (4). In this case, the optimization is an instance of semidefinite programming [18]:

$$\begin{aligned} \min \quad & \textstyle\sum_c \operatorname{trace}(\mathbf{\Psi}_c) \\ \text{s.t.} \quad & 1 + \boldsymbol{z}_n^{\mathrm{T}}(\mathbf{\Phi}_{y_n} - \mathbf{\Phi}_c)\boldsymbol{z}_n \leq 0, \quad \forall c \neq y_n, n = 1, 2, \ldots, N \\ & \mathbf{\Phi}_c \succ 0, \quad c = 1, 2, \ldots, C \end{aligned} \tag{5}$$

Note that the trace of the matrix $\mathbf{\Psi}_c$ appears in the above objective function, as opposed to the trace of the matrix $\mathbf{\Phi}_c$, as defined in eq. (2); minimizing the former imposes the scale regularization only on the inverse covariance matrices of the GMM, while the latter would improperly regularize the mean vectors as well. The constraints $\mathbf{\Phi}_c \succ 0$ restrict the matrices to be positive semidefinite.

The objective function must be modified for training data that lead to infeasible constraints in eq. (5). As in SVMs, we introduce nonnegative slack variables $\xi_{nc}$ to monitor the amount by which the margin constraints in eq. (4) are violated [15]. The objective function in this setting balances the margin violations versus the scale regularization:

$$\begin{aligned} \min \quad & \textstyle\sum_{nc} \xi_{nc} + \gamma \sum_c \operatorname{trace}(\mathbf{\Psi}_c) \\ \text{s.t.} \quad & 1 + \boldsymbol{z}_n^{\mathrm{T}}(\mathbf{\Phi}_{y_n} - \mathbf{\Phi}_c)\boldsymbol{z}_n \leq \xi_{nc}, \\ & \xi_{nc} \geq 0, \quad \forall c \neq y_n, n = 1, 2, \ldots, N \\ & \mathbf{\Phi}_c \succ 0, \quad c = 1, 2, \ldots, C \end{aligned} \tag{6}$$

where the balancing hyperparameter $\gamma > 0$ is set by some form of cross-validation. This optimization is also an instance of semidefinite programming.

## 2.3 Softmax margin maximization for multiple mixture components

Lastly we review the extension to mixture modeling where each class is represented by multiple ellipsoids [15]. Let $\mathbf{\Phi}_{cm}$ denote the matrix for the $m^{\text{th}}$ ellipsoid (or mixture component) in class $c$. We imagine that each example $\boldsymbol{x}_n$ has not only a class label $y_n$, but also a mixture component label $m_n$. Such labels are not provided a priori in the training data, but we can generate "proxy" labels by fitting GMMs to the examples in each class by maximum likelihood estimation, then for each example, computing the mixture component with the highest posterior probability.

In the setting where each class is represented by multiple ellipsoids, the goal of learning is to ensure that each example is closer to its "target" ellipsoid than the ellipsoids from all other *classes*. Specifically, for a labeled example $(\boldsymbol{x}_n, y_n, m_n)$, the constraint in eq. (4) is replaced by the $M$ constraints:

$$\forall c \neq y_n, \ \forall m, \quad \boldsymbol{z}_n^{\mathrm{T}}(\mathbf{\Phi}_{cm} - \mathbf{\Phi}_{y_n m_n})\boldsymbol{z}_n \geq 1, \tag{7}$$

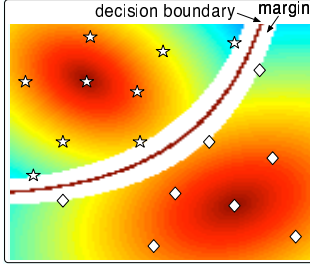

Figure 1: Decision boundary in a large margin GMM: labeled examples lie at least one unit of distance away.

| mixture | EM | margin |
|---|---|---|
| 1 | 4.2% | 1.4% |
| 2 | 3.4% | 1.4% |
| 4 | 3.0% | 1.2% |
| 8 | 3.3% | 1.5% |

Table 1: Test error rates on MNIST digit recognition: maximum likelihood versus large margin GMMs.

where $M$ is the number of mixture components (assumed, for simplicity, to be the same for each class). We fold these multiple constraints into a single one by appealing to the "softmax" inequality: $\min_m a_m \geq -\log \sum_m e^{-a_m}$. Specifically, using the inequality to derive a lower bound on $\min_m \boldsymbol{z}_n^{\mathrm{T}} \boldsymbol{\Phi}_{cm} \boldsymbol{z}_n$, we replace the $M$ constraints in eq. (7) by the stricter constraint:

$$\forall c \neq y_n, \quad -\log \sum_m e^{-\boldsymbol{z}_n^{\mathrm{T}} \boldsymbol{\Phi}_{cm} \boldsymbol{z}_n} - \boldsymbol{z}_n^{\mathrm{T}} \boldsymbol{\Phi}_{y_n m_n} \boldsymbol{z}_n \geq 1. \tag{8}$$

We will use a similar technique in section 3 to handle the exponentially many constraints that arise in sequential classification. Note that the inequality in eq. (8) implies the inequality of eq. (7) but not vice versa. Also, though nonlinear in the matrices $\{\boldsymbol{\Phi}_{cm}\}$, the constraint in eq. (8) is still convex.

The objective function in eq. (6) extends straightforwardly to this setting. It balances a regularizing term that sums over ellipsoids versus a penalty term that sums over slack variables, one for each constraint in eq. (8). The optimization is given by:

$$\begin{aligned}
\min \quad & \sum_{nc} \xi_{nc} + \gamma \sum_{cm} \mathrm{trace}(\boldsymbol{\Psi}_{cm}) \\
\text{s.t.} \quad & 1 + \boldsymbol{z}_n^{\mathrm{T}} \boldsymbol{\Phi}_{y_n m_n} \boldsymbol{z}_n + \log \sum_m e^{-\boldsymbol{z}_n^{\mathrm{T}} \boldsymbol{\Phi}_{cm} \boldsymbol{z}_n} \leq \xi_{nc}, \\
& \xi_{nc} \geq 0, \quad \forall c \neq y_n, n = 1, 2, \ldots, N \\
& \boldsymbol{\Phi}_{cm} \succ 0, \quad c = 1, 2, \ldots, C, \ m = 1, 2, \ldots, M
\end{aligned} \tag{9}$$

This optimization is not an instance of semidefinite programming, but it is convex. We discuss how to perform the optimization efficiently for large data sets in appendix A.

## 2.4 Handwritten digit recognition

We trained large margin GMMs for multiway classification of MNIST handwritten digits [8]. The MNIST data set has 60000 training examples and 10000 test examples. Table 1 shows that the large margin GMMs yielded significantly lower test error rates than GMMs trained by maximum likelihood estimation. Our best results are comparable to the best SVM results (1.0-1.4%) on deskewed images [8] that do not make use of prior knowledge. For our best model, with four mixture components per digit class, the core training optimization over all training examples took five minutes on a PC. (Multiple runs of this optimization on smaller validation sets, however, were also required to set two hyperparameters: the regularizer for model complexity, and the termination criterion for early stopping.)

## 3 Large margin HMMs for sequential classification

In this section, we extend the framework in the previous section from multiway classification to sequential classification. Particularly, we have in mind the application to ASR, where GMMs are used to parameterize the emission densities of CD-HMMs. Strictly speaking, the GMMs in our framework cannot be interpreted as emission densities because their parameters are not constrained to represent normalized distributions. Such an interpretation, however, is not necessary for their use as discriminative models. In sequential classification by CD-HMMs, the goal is to infer the correct hidden state sequence $\boldsymbol{y} = [y_1, y_2, \ldots, y_T]$ given the observation sequence $\boldsymbol{X} = [\boldsymbol{x}_1, \boldsymbol{x}_2, \ldots, \boldsymbol{x}_T]$. In the application to ASR, the hidden states correspond to phoneme labels, and the observations are

acoustic feature vectors. Note that if an observation sequence has length $T$ and each label can belong to $C$ classes, then the number of incorrect state sequences grows as $O(C^T)$. This combinatorial explosion presents the main challenge for large margin methods in sequential classification: how to separate the correct hidden state sequence from the exponentially large number of incorrect ones.

The section is organized as follows. Section 3.1 explains the way that margins are computed for sequential classification. Section 3.2 describes our algorithm for large margin training of CD-HMMs. Details are given only for the simple case where the observations in each hidden state are modeled by a single ellipsoid. The extension to multiple mixture components closely follows the approach in section 2.3 and can be found in [14, 16]. Margin-based learning of transition probabilities is likewise straightforward but omitted for brevity. Both these extensions were implemented, however, for the experiments on phonetic recognition in section 3.3.

## 3.1   Margin constraints for sequential classification

We start by defining a discriminant function over state (label) sequences of the CD-HMM. Let $a(i, j)$ denote the transition probabilities of the CD-HMM, and let $\boldsymbol{\Phi}_s$ denote the ellipsoid parameters of state $s$. The discriminant function $\mathcal{D}(\boldsymbol{X}, \boldsymbol{s})$ computes the score of the state sequence $\boldsymbol{s} = [s_1, s_2, \dots, s_T]$ on an observation sequence $\boldsymbol{X} = [\boldsymbol{x}_1, \boldsymbol{x}_2, \dots, \boldsymbol{x}_T]$ as:

$$\mathcal{D}(\boldsymbol{X}, \boldsymbol{s}) = \sum_t \log a(s_{t-1}, s_t) - \sum_{t=1}^{T} \boldsymbol{z}_t^{\mathrm{T}} \boldsymbol{\Phi}_{s_t} \boldsymbol{z}_t. \tag{10}$$

This score has the same form as the log-probability $\log P(\boldsymbol{X}, \boldsymbol{s})$ in a CD-HMM with Gaussian emission densities. The first term accumulates the log-transition probabilities along the state sequence, while the second term accumulates "acoustic scores" computed as the Mahalanobis distances to each state's centroid. In the setting where each state is modeled by multiple mixture components, the acoustic scores from individual Mahalanobis distances are replaced with "softmax" distances of the form $\log \sum_{m=1}^{M} e^{-\boldsymbol{z}_t^{\mathrm{T}} \boldsymbol{\Phi}_{s_t m} \boldsymbol{z}_t}$, as described in section 2.3 and [14, 16].

We introduce margin constraints in terms of the above discriminant function. Let $\mathcal{H}(\boldsymbol{s}, \boldsymbol{y})$ denote the Hamming distance (i.e., the number of mismatched labels) between an arbitrary state sequence $\boldsymbol{s}$ and the target state sequence $\boldsymbol{y}$. Earlier, in section 2 on multiway classification, we constrained each labeled example to lie at least one unit distance from the decision boundary to each competing class; see eq. (4). Here, by extension, we constrain the score of each target sequence to exceed that of each competing sequence by an amount equal to or greater than the Hamming distance:

$$\forall \boldsymbol{s} \neq \boldsymbol{y}, \quad \mathcal{D}(\boldsymbol{X}, \boldsymbol{y}) - \mathcal{D}(\boldsymbol{X}, \boldsymbol{s}) \geq \mathcal{H}(\boldsymbol{s}, \boldsymbol{y}) \tag{11}$$

Intuitively, eq. (11) requires that the (log-likelihood) gap between the score of an incorrect sequence $\boldsymbol{s}$ and the target sequence $\boldsymbol{y}$ should grow in proportion to the number of individual label errors. The appropriateness of such proportional constraints for sequential classification was first noted by [17].

## 3.2   Softmax margin maximization for sequential classification

The challenge of large margin sequence classification lies in the exponentially large number of constraints, one for each incorrect sequence $\boldsymbol{s}$, embodied by eq. (11). We will use the same softmax inequality, previously introduced in section 2.3, to fold these multiple constraints into one, thus considerably simplifying the optimization required for parameter estimation. We first rewrite the constraint in eq. (11) as:

$$-\mathcal{D}(\boldsymbol{X}, \boldsymbol{y}) + \max_{\boldsymbol{s} \neq \boldsymbol{y}} \{\mathcal{H}(\boldsymbol{s}, \boldsymbol{y}) + \mathcal{D}(\boldsymbol{X}, \boldsymbol{s})\} \leq 0 \tag{12}$$

We obtain a more manageable constraint by substituting a softmax upper bound for the $\max$ term and requiring that the inequality still hold:

$$-\mathcal{D}(\boldsymbol{X}, \boldsymbol{y}) + \log \sum_{\boldsymbol{s} \neq \boldsymbol{y}} e^{\mathcal{H}(\boldsymbol{s}, \boldsymbol{y}) + \mathcal{D}(\boldsymbol{X}, \boldsymbol{s})} \leq 0 \tag{13}$$

Note that eq. (13) implies eq. (12) but not vice versa. As in the setting for multiway classification, the objective function for sequential classification balances two terms: one regularizing the scale of

the GMM parameters, the other penalizing margin violations. Denoting the training sequences by $\{\boldsymbol{X}_n, \boldsymbol{y}_n\}_{n=1}^N$ and the slack variables (one for each training sequence) by $\xi_n \geq 0$, we obtain the following convex optimization:

$$
\begin{array}{ll}
\min & \sum_n \xi_n + \gamma \sum_{cm} \text{trace}(\boldsymbol{\Psi}_{cm}) \\
\text{s.t.} & -\mathcal{D}(\boldsymbol{X}_n, \boldsymbol{y}_n) + \log \sum_{\boldsymbol{s} \neq \boldsymbol{y}_n} e^{\mathcal{H}(\boldsymbol{s}, \boldsymbol{y}_n) + \mathcal{D}(\boldsymbol{X}_n, \boldsymbol{s})} \leq \xi_n, \\
& \xi_n \geq 0, \quad n = 1, 2, \ldots, N \\
& \boldsymbol{\Phi}_{cm} \succ 0, \quad c = 1, 2, \ldots, C, m = 1, 2, \ldots, M
\end{array}
\tag{14}
$$

It is worth emphasizing several crucial differences between this optimization and previous ones [4, 11, 20] for discriminative training of CD-HMMs for ASR. First, the softmax large margin constraint in eq. (13) is a differentiable function of the model parameters, as opposed to the "hard" maximum in eq. (12) and the number of classification errors in the MCE training criteria [4]. The constraint and its gradients with respect to GMM parameters $\boldsymbol{\Phi}_{cm}$ and transition parameters $a(\cdot, \cdot)$ can be computed efficiently using dynamic programming, by a variant of the standard forward-backward procedure in HMMs [14]. Second, due to the reparameterization in eq. (2), the discriminant function $\mathcal{D}(\boldsymbol{X}_n, \boldsymbol{y}_n)$ and the softmax function are convex in the model parameters. Therefore, the optimization eq. (14) can be cast as convex optimization, avoiding spurious local minima [14]. Third, the optimization not only increases the log-likelihood gap between correct and incorrect state sequences, but also drives the gap to grow in proportion to the number of individually incorrect labels (which we believe leads to more robust generalization). Finally, compared to the large margin framework in [17], the softmax handling of exponentially large number of margin constraints makes it possible to train on larger data sets. We discuss how to perform the optimization efficiently in appendix A.

### 3.3 Phoneme recognition

We used the TIMIT speech corpus [7, 9, 12] to perform experiments in phonetic recognition. We followed standard practices in preparing the training, development, and test data. Our signal processing front-end computed 39-dimensional acoustic feature vectors from 13 mel-frequency cepstral coefficients and their first and second temporal derivatives. In total, the training utterances gave rise to roughly 1.2 million frames, all of which were used in training.

We trained baseline maximum likelihood recognizers and two different types of large margin recognizers. The large margin recognizers in the first group were "low-cost" discriminative CD-HMMs whose GMMs were merely trained for frame-based classification. In particular, these GMMs were estimated by solving the optimization in eq. (8), then substituted into first-order CD-HMMs for sequence decoding. The large margin recognizers in the second group were fully trained for sequential classification. In particular, their CD-HMMs were estimated by solving the optimization in eq. (14), generalized to multiple mixture components and adaptive transition parameters [14, 16]. In all the recognizers, the acoustic feature vectors were labeled by 48 phonetic classes, each represented by one state in a first-order CD-HMM.

For each recognizer, we compared the phonetic state sequences obtained by Viterbi decoding to the "ground-truth" phonetic transcriptions provided by the TIMIT corpus. For the purpose of computing error rates, we followed standard conventions in mapping the 48 phonetic state labels down to 39 broader phone categories. We computed two different types of phone error rates, one based on Hamming distance, the other based on edit distance. The former was computed simply from the percentage of mismatches at the level of individual frames. The latter was computed by aligning the Viterbi and ground truth transcriptions using dynamic programming [9] and summing the substitution, deletion, and insertion error rates from the alignment process. The "frame-based" phone error rate computed from Hamming distances is more closely tracked by our objective function for large margin training, while the "string-based" phone error rate computed from edit distances provides a more relevant metric for ASR.

Tables 2 and 3 show the results of our experiments. For both types of error rates, and across all model sizes, the best performance was consistently obtained by large margin CD-HMMs trained for sequential classification. Moreover, among the two different types of large margin recognizers, utterance-based training generally yielded significant improvement over frame-based training.

Discriminative learning of CD-HMMs is an active research area in ASR. Two types of algorithms have been widely used: maximum mutual information (MMI) [20] and minimum classification er-

| mixture (per state) | baseline (EM) | margin (frame) | margin (utterance) |
|---|---|---|---|
| 1 | 45% | 37% | 30% |
| 2 | 45% | 36% | 29% |
| 4 | 42% | 35% | 28% |
| 8 | 41% | 34% | 27% |

| mixture (per state) | baseline (EM) | margin (frame) | margin (utterance) |
|---|---|---|---|
| 1 | 40.1% | 36.3% | 31.2% |
| 2 | 36.5% | 33.5% | 30.8% |
| 4 | 34.7% | 32.6% | 29.8% |
| 8 | 32.7% | 31.0% | 28.2% |

Table 2: Frame-based phone error rates, from Hamming distance, of different recognizers. See text for details.

Table 3: String-based phone error rates, from edit distance, of different recognizers. See text for details.

ror [4]. In [16], we compare the large margin training proposed in this paper to both MMI and MCE systems for phoneme recognition trained on the exact same acoustic features. There we find that the large margin approach leads to lower error rates, owing perhaps to the absence of local minima in the objective function and/or the use of margin constraints based on Hamming distances.

## 4 Discussion

Discriminative learning of sequential models is an active area of research in both ASR [10, 13, 20] and machine learning [1, 6, 17]. This paper makes contributions to lines of work in both communities. First, in distinction to previous work in ASR, we have proposed a convex, margin-based cost function that penalizes incorrect decodings in proportion to their Hamming distance from the desired transcription. The use of the Hamming distance in this context is a crucial insight from the work of [17] in the machine learning community, and it differs profoundly from merely penalizing the log-likelihood gap between incorrect and correct transcriptions, as commonly done in ASR.

Second, in distinction to previous work in machine learning, we have proposed a framework for sequential classification that naturally integrates with the infrastructure of modern speech recognizers. Using the softmax function, we have also proposed a novel way to monitor the exponentially many margin constraints that arise in sequential classification. For real-valued observation sequences, we have shown how to train large margin HMMs via convex optimizations over their parameter space of positive semidefinite matrices. Finally, we have demonstrated that these learning algorithms lead to improved sequential classification on data sets with over one million training examples (i.e., phonetically labeled frames of speech). In ongoing work, we are applying our approach to large vocabulary ASR and other tasks such as speaker identification and visual object recognition.

## A Solver

The optimizations in eqs. (5), (6), (9) and (14) are convex: specifically, in terms of the matrices that parameterize large margin GMMs and HMMs, the objective functions are linear, while the constraints define convex sets. Despite being convex, however, these optimizations cannot be managed by off-the-shelf numerical optimization solvers or generic interior point methods for problems as large as the ones in this paper. We devised our own special-purpose solver for these purposes.

For simplicity, we describe our solver for the optimization of eq. (6), noting that it is easily extended to eqs. (9) and (14). To begin, we eliminate the slack variables and rewrite the objective function in terms of the hinge loss function: $\text{hinge}(z) = \max(0, z)$. This yields the objective function:

$$\mathcal{L} = \sum_{n, c \neq y_n} \text{hinge}\left(1 + z_n^{\text{T}}(\Phi_{y_n} - \Phi_c)z_n\right) + \gamma \sum_c \text{trace}(\Psi_c), \qquad (15)$$

which is convex in terms of the positive semidefinite matrices $\Phi_c$. We minimize $\mathcal{L}$ using a projected subgradient method [2], taking steps along the subgradient of $\mathcal{L}$, then projecting the matrices $\{\Phi_c\}$ back onto the set of positive semidefinite matrices after each update. This method is guaranteed to converge to the global minimum, though it typically converges very slowly. For faster convergence, we precede this method with an unconstrained conjugate gradient optimization in the square-root matrices $\{\Omega_c\}$, where $\Phi_c = \Omega_c \Omega_c^{\text{T}}$. The latter optimization is not convex, but in practice it rapidly converges to an excellent starting point for the projected subgradient method.

## Acknowledgment

This work was supported by the National Science Foundation under grant Number 0238323. We thank F. Pereira, K. Crammer, and S. Roweis for useful discussions and correspondence. Part of this work was conducted while both authors were affiliated with the University of Pennsylvania.

## References

[1] Y. Altun, I. Tsochantaridis, and T. Hofmann. Hidden markov support vector machines. In T. Fawcett and N. Mishra, editors, *Proceedings of the Twentieth International Conference (ICML 2003)*, pages 3–10, Washington, DC, USA, 2003. AAAI Press.

[2] D. P. Bertsekas. *Nonlinear programming*. Athena Scientific, 2nd edition, 1999.

[3] P. S. Gopalakrishnan, D. Kanevsky, A. Nádas, and D. Nahamoo. An inequality for rational functions with applications to some statistical estimation problems. *IEEE Trans. Info. Theory*, 37(1):107—113, 1991.

[4] B.-H. Juang and S. Katagiri. Discriminative learning for minimum error classification. *IEEE Trans. Sig. Proc.*, 40(12):3043–3054, 1992.

[5] S. Kapadia, V. Valtchev, and S. Young. MMI training for continuous phoneme recognition on the TIMIT database. In *Proc. of ICASSP 93*, volume 2, pages 491–494, Minneapolis, MN, 1993.

[6] J. Lafferty, A. McCallum, and F. C. N. Pereira. Conditional random fields: Probabilisitc models for segmenting and labeling sequence data. In *Proc. 18th International Conf. on Machine Learning (ICML 2001)*, pages 282–289. Morgan Kaufmann, San Francisco, CA, 2001.

[7] L. F. Lamel, R. H. Kassel, and S. Seneff. Speech database development: design and analsysis of the acoustic-phonetic corpus. In L. S. Baumann, editor, *Proceedings of the DARPA Speech Recognition Workshop*, pages 100–109, 1986.

[8] Y. LeCun, L. Jackel, L. Bottou, A. Brunot, C. Cortes, J. Denker, H. Drucker, I. Guyon, U. Muller, E. Sackinger, P. Simard, and V. Vapnik. Comparison of learning algorithms for handwritten digit recognition. In F. Fogelman and P. Gallinari, editors, *Proceedings of the International Conference on Artificial Neural Networks*, pages 53–60, 1995.

[9] K. F. Lee and H. W. Hon. Speaker-independent phone recognition using hidden Markov models. *IEEE Transactions on Acoustics, Speech, and Signal Processing*, 37(11):1641–1648, 1988.

[10] X. Li, H. Jiang, and C. Liu. Large margin HMMs for speech recognition. In *Proceedings of ICASSP 2005*, pages 513–516, Philadelphia, 2005.

[11] A. Nádas. A decision-theoretic formulation of a training problem in speech recognition and a comparison of training by unconditional versus conditional maximum likelihood. *IEEE Transactions on Acoustics, Speech and Signal Processing*, 31(4):814–817, 1983.

[12] T. Robinson. An application of recurrent nets to phone probability estimation. *IEEE Transactions on Neural Networks*, 5(2):298–305, 1994.

[13] J. L. Roux and E. McDermott. Optimization methods for discriminative training. In *Proceedings of Nineth European Conference on Speech Communication and Technology (EuroSpeech 2005)*, pages 3341–3344, Lisbon, Portgual, 2005.

[14] F. Sha. *Large margin training of acoustic models for speech recognition*. PhD thesis, University of Pennsylvania, 2007.

[15] F. Sha and L. K. Saul. Large margin Gaussian mixture modeling for phonetic classification and recognition. In *Proceedings of ICASSP 2006*, pages 265–268, Toulouse, France, 2006.

[16] F. Sha and L. K. Saul. Comparison of large margin training to other discriminative methods for phonetic recognition by hidden Markov models. In *Proceedings of ICASSP 2007*, Hawaii, 2007.

[17] B. Taskar, C. Guestrin, and D. Koller. Max-margin markov networks. In S. Thrun, L. Saul, and B. Schölkopf, editors, *Advances in Neural Information Processing Systems (NIPS 16)*. MIT Press, Cambridge, MA, 2004.

[18] L. Vandenberghe and S. P. Boyd. Semidefinite programming. *SIAM Review*, 38(1):49–95, March 1996.

[19] V. Vapnik. *Statistical Learning Theory*. Wiley, N.Y., 1998.

[20] P. C. Woodland and D. Povey. Large scale discriminative training of hidden Markov models for speech recognition. *Computer Speech and Language*, 16:25–47, 2002.

[21] S. J. Young. Acoustic modelling for large vocabulary continuous speech recognition. In K. Ponting, editor, *Computational Models of Speech Pattern Processing*, pages 18–39. Springer, 1999.
